# An Analog VLSI Chip for Radial Basis Functions

Janeen Anderson

John C. Platt
Synaptics, Inc.
2698 Orchard Parkway
San Jose, CA 95134

David B. Kirk*

## Abstract

We have designed, fabricated, and tested an analog VLSI chip which computes radial basis functions in parallel. We have developed a synapse circuit that approximates a quadratic function. We aggregate these circuits to form radial basis functions. These radial basis functions are then averaged together using a follower aggregator.

## 1  INTRODUCTION

Radial basis functions (RBFs) are a method for approximating a function from scattered training points [Powell, 1987]. RBFs have been used to solve recognition and prediction problems with a fair amount of success [Lee, 1991] [Moody, 1989] [Platt, 1991]. The first layer of an RBF network computes the distance of the input to the network to a set of stored memories. Each basis function is a non-linear function of a corresponding distance. The basis functions are then added together with second-layer weights to produce the output of the network. The general form of an RBF is

$$y_i = \sum_j h_{ij} \phi_j \left( \|\vec{I} - \vec{c}_j\| \right), \qquad (1)$$

where $y_i$ is the output of the network, $h_{ij}$ is the second-layer weight, $\phi_j$ is the non-linearity, $\vec{c}_j$ is the $j$th memory stored in the network and $\vec{I}$ is the input to

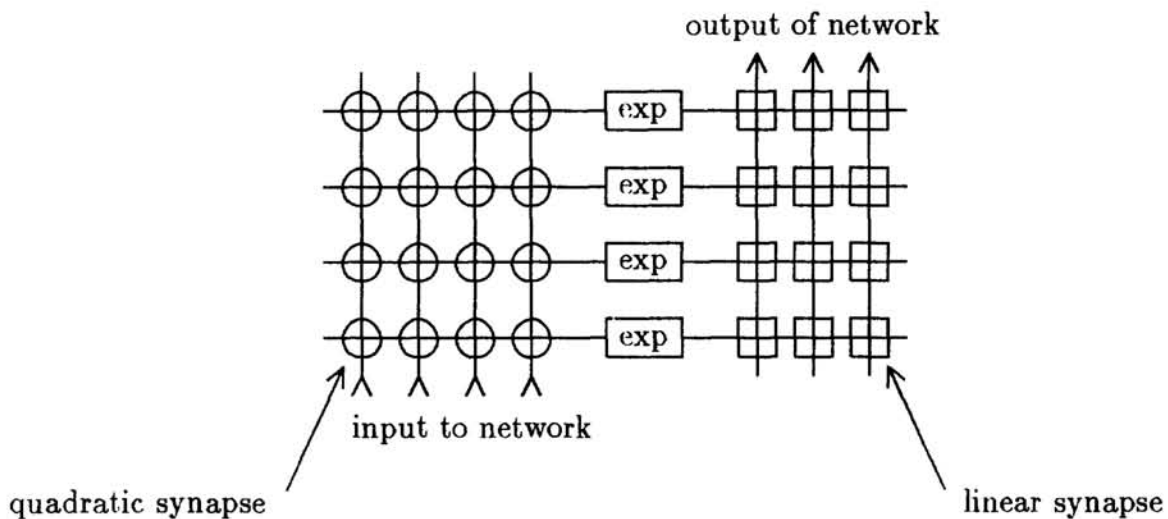

Figure 1: The architecture of a Gaussian RBF network.

the network. Many researchers use Gaussians to create basis functions that have a localized effect in input space [Poggio, 1990][Moody, 1989]:

$$y_i = \sum_j h_{ij} \exp\left(-\frac{1}{2\sigma^2}\sum_k (I_k - c_{jk})^2\right). \qquad (2)$$

The architecture of a Gaussian RBF network is shown in figure 1.

RBFs can be implemented either via software or hardware. If high speed is not necessary, then computing all of the basis functions in software is adequate. However, if an application requires many inputs or high speed, then hardware is required.

RBFs use a lot of operations more complex than simply multiplication and addition. For example, a Gaussian RBF requires an exponential for every basis function. Using a partition of unity requires a divide for every basis function. Analog VLSI is an attractive way of computing these complex operations very quickly: we can compute all of the basis functions in parallel, using a few transistors per synapse.

This paper discusses an analog VLSI chip that computes radial basis functions. We discuss how we map the mathematical model of an RBF into compact analog hardware. We then present results from a test chip that was fabricated. We discuss possible applications for the hardware architecture and future theoretical work.

## 2   MAPPING RADIAL BASIS FUNCTIONS INTO HARDWARE

In order to create an analog VLSI chip, we must map the idea of radial basis functions into transistors. In order to create a high-density chip, the mathematics of RBFs must be modified to be computed more naturally by transistor physics. This section discusses the mapping from Gaussian RBFs into CMOS circuitry.

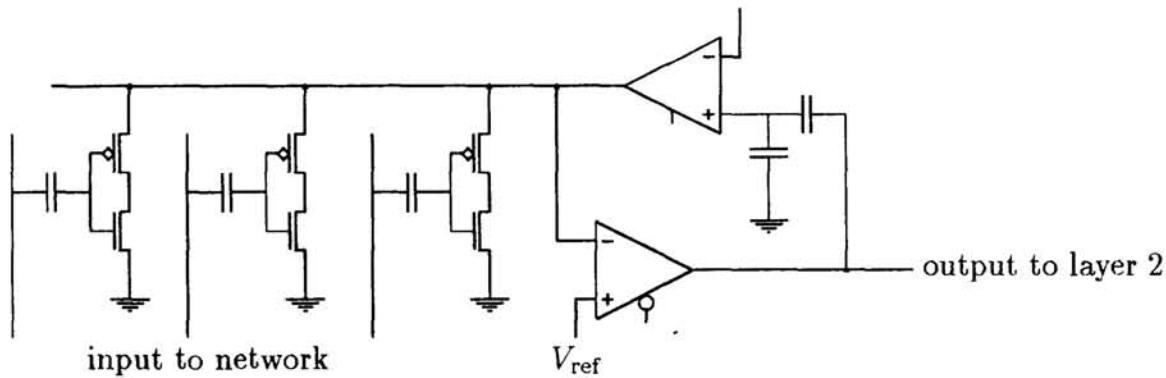

Figure 2: Circuit diagram for first-layer neuron, showing three Gaussian synapses and the sense amplifier.

## 2.1  Computing Quadratic Distance

Ideally, the first-layer synapses in figure 1 would compute a quadratic distance of the input to a stored value. Quadratics go to infinity for large values of their input, hence are hard to build in analog hardware and are not robust against outliers in the input data. Therefore, it is much more desirable to use a saturating non-linearity: we will use a Gaussian for a first-layer synapse, which approximates a quadratic near its peak.

We implement the first-layer Gaussian synapse using an inverter (see figure 2). The current running through each inverter from the voltage rail to ground is a Gaussian function of the inverter's input, with the peak of the Gaussian occurring halfway between the voltage rail and ground [Mead, 1980][Mead, 1992].

To adjust the center of the Gaussian, we place a capacitor between the input to the synapse and the input of the inverter. The inverter thus has a floating gate input. We adjust the charge on the floating gate by using a combination of tunneling and non-avalanche hot electron injection [Anderson, 1990] [Anderson, 1992].

All of the Gaussian synapses for one neuron share a voltage rail. The sense amplifier holds that voltage rail at a particular voltage, $V_{ref}$. The output of the sense amplifier is a voltage which is linear in the total current being drawn by the Gaussian synapses. We use a floating gate in the sense amplifier to ensure that the output of the sense amplifier is known when the input to the network is at a known state. Again, we adjust the floating gate via tunneling and injection.

Figure 3 shows the output of the sense amplifier for four different neurons. The data was taken from a real chip, described in section 3. The figure shows that the top of a Gaussian approximates a quadratic reasonably well. Also, the width and heights of the outputs of each first-layer neuron match very well, because the circuit is operated above threshold.

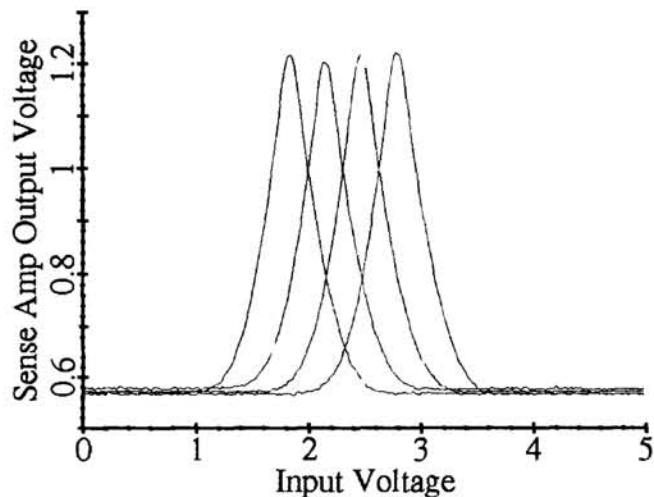

Figure 3: Measured output of set of four first-layer neurons. All of the synapses of each neuron are programmed to peak at the same voltage. The $x$-axis is the input voltage, and the $y$-axis is the voltage output of the sense amplifier

## 2.2   Computing the Basis Function

To compute a Gaussian basis function, the distance produced by the first layer needs to be exponentiated. Since the output of the sense amplifier is a voltage negatively proportional to the distance, a subthreshold transistor can perform this exponentiation.

However, subthreshold circuits can be slow. Also, the choice of a Gaussian basis function is somewhat arbitrary [Poggio, 1990]. Therefore, we choose to adjust the sense amplifier to produce a voltage that is both above and below threshold. The basis function that the chip computes can be expressed as

$$S_j = \sum_k \text{Gaussian}(I_k - c_{jk}); \tag{3}$$

$$\phi_j = \begin{cases} (S_j - \theta)^2, & \text{if } S_j > \theta; \\ 0, & \text{otherwise.} \end{cases} \tag{4}$$

where $\theta$ is a threshold that is set by how much current is required by the sense amplifier to produce an output equal to the threshold voltage of a N-type transistor.

Equations 3 and 4 have an intuitive explanation. Each first-layer synapse votes on whether its input matched its stored value. The sum of these votes is $S_j$. If the sum $S_j$ is less than a threshold $\theta$, then the basis function $\phi_j$ is zero. However, if the number of votes exceeds the threshold, then the basis function turns on. Therefore, one can adjust the dimensionality of the basis function by adjusting $\theta$: the dimensionality is $\lceil N - \theta - 1 \rceil$, where $N$ is the number of inputs to the network.

Figure 4 shows how varying $\theta$ changes the basis function, for $N = 2$. The input to the network is a two-dimensional space, represented by location on the page. The value of the basis function is represented by the darkness of the ink. Setting $\theta = 1$ yields the basis function on the left, which is a fuzzy 0-dimensional point. Setting $\theta = 0$ yields the basis function on the right, which is a union of fuzzy 1-dimensional lines.

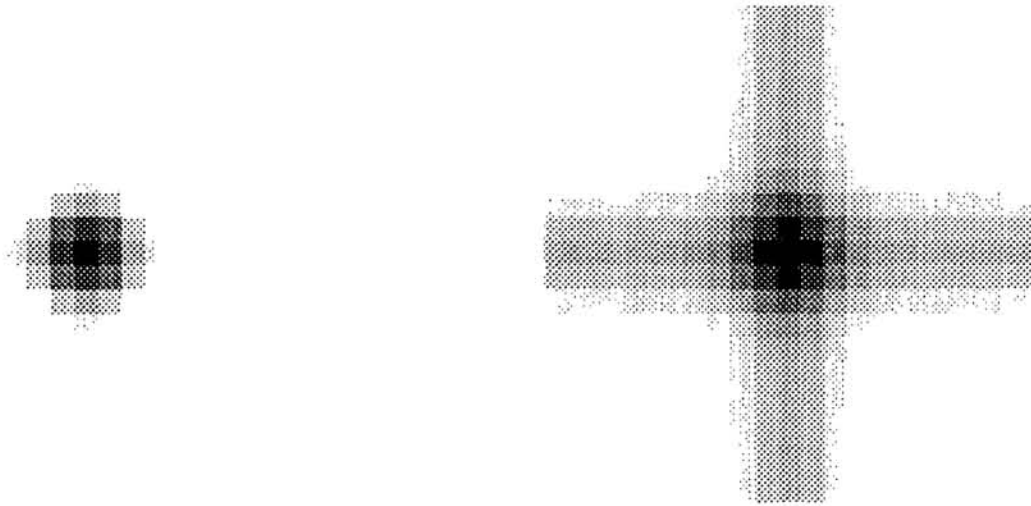

Figure 4: Examples of two simulated basis functions with differing dimensionality.

Having an adjustable dimension for basis functions is useful, because it increases the robustness of the basis function. A Gaussian radial basis function is non-zero only when all elements of the input vector roughly match the center of the Gaussian. By using a hardware basis function, we can allow certain inputs not to match, while still turning on the basis function.

## 2.3   Blending the Basis Functions

To make the blending of the basis functions easier to implement in analog VLSI, we decided to use an alternative method for basis function combination, called the partition of unity [Moody, 1989]:

$$y_i = \frac{\sum_j h_{ij} \phi_j}{\sum_j \phi_j}. \tag{5}$$

The partition of unity suggests that the second layer should compute a weighted average of first-layer outputs, not just a weighted sum. We can compute a weighted average reasonably well with a follower aggregator used in the linear region [Mead, 1989].

Equations 4 and 5 can both be implemented by using a wide-range amplifier as a synapse (see figure 5). The bias of the amplifier is the output of the sense amplifier. That way, the above-threshold non-linearity of the bias transistor is applied to the output of the first layer and implements equation 4. The amplifier then attempts to drag the output of the second-layer neuron towards a stored value $h_{ij}$ and implements equation 5. We store the value on a floating gate, using tunneling and injection.

The follower aggregator does not implement equation 5 perfectly: the amplifiers saturate, hence introduce a non-linearity. A follower aggregator implements

$$\sum_j \tanh(\alpha(h_{ij} - y_i))\phi_j = 0. \tag{6}$$

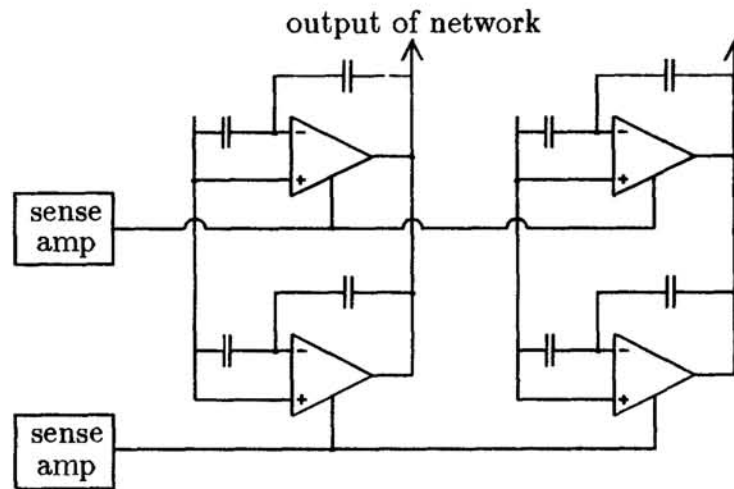

Figure 5: Circuit diagram for second-layer synapses.

We use a capacitive divider to increase the linear range (decrease $\alpha$) of the amplifiers. However, the non-linearity of the amplifiers may be beneficial, because it reduces the effect of outliers in the stored $h_{ij}$ values.

## 3  RESULTS

We fabricated the chip in 2 micron CMOS. The current version of the chip has 8 inputs, 159 basis functions and 4 outputs. The chip size is 2.2 millimeters by 9.6 millimeters

The core radial basis function circuitry works end-to-end. By measuring the output of the sense amplifier, we can measure the response of the first layer, which is shown in figure 3. Experiments show that the average width of the first-layer Gaussians is 0.350 volts, with a standard deviation of 23 millivolts. The centers of the first-layer Gaussians can be programmed more accurately than 15 millivolts, which is the resolution of the test setup for this chip. Further experiments show that the second-layer followers are linear to within 4% over 5 volts. Due to one mis-sized transistor, programming the second layer accurately is difficult.

We have successfully tested the chip at 90 kHz, which is the speed limit of the current test setup. We have not yet tested the chip at its full speed. The static power dissipation of the chip is 2 milliwatts.

Figure 6 shows an example of real end-to-end output of the chip. All synapses for each first-layer neuron are programmed to the same value. The first-layer neurons are programmed to a ramp: each neuron is programmed to respond to a voltage 32 millivolts higher than the previous neuron. The second layer neurons are programmed to values shown by $y$-values of the dots in figure 6. The output of the chip is shown as the solid line in figure 6. The output is measured as all of the inputs to the chip are swept simultaneously. The chip splines and smooths out the noisy stored second-layer values. Notice that the stored second-layer values are low for inputs near 2.5 V: the output of a chip is correspondingly lower.

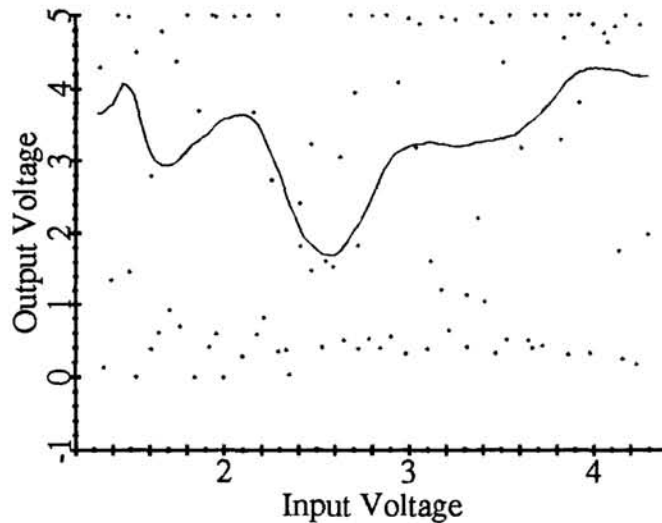

Figure 6: Example of end-to-end output measured from the chip.

# 4   FUTURE WORK

The mathematical model of the hardware network suggests interesting theoretical future work. There are two novel features of this model: the variable dimensionality of the basis functions, and the non-linearity in the partition of unity. More simulation work needs to be done to see how much of an benefit these features yield.

The chip architecture discussed in this paper is suitable for many medium-dimensional function mapping problems where radial basis functions are appropriate. For example, the chip is useful for high speed control, optical character recognition, and robotics.

One application of the chip we have studied further is the antialiasing of printed characters, with proportional spacing, multiple fonts, and arbitrary scaling. Each antialiased pixel has an intensity which is the integral of the character's partial coverage of that pixel convolved with some filter. The chip could perform a function interpolation for each pixel of each character. The function being interpolated is the intensity integral, based on the subpixel coverage as convolved with the antialiasing filter kernel. Figure 7 shows the results of the anti-aliasing of the character using a simulation of the chip.

# 5   CONCLUSIONS

We have described a multi-layer analog VLSI neural network chip that computes radial basis functions in parallel. We use inverters as first-layer synapses, to compute Gaussians that approximate quadratics. We use follower aggregators as second-layer neurons, to compute the basis functions and to blend the basis functions using a partition of unity. Preliminary experiments with a test chip shows that the core radial basis function circuitry works. In the future, we will explore the new basis function model suggested by the hardware and further investigate applications of the chip.

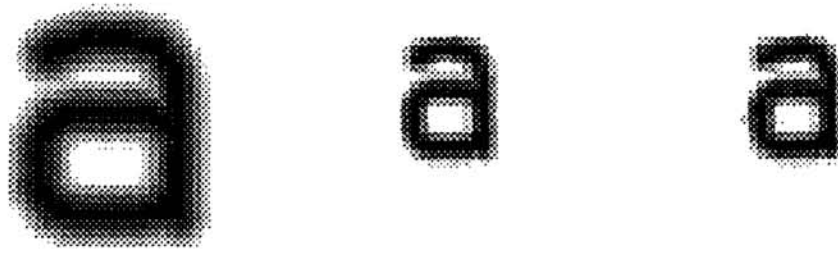

Figure 7: Three images of the letter "a". The image on the left is the high resolution anti-aliased version of the character. The middle image is a smaller version of the left image. The right image is the chip simulation, trained to be close to the middle image, by using the left image as the training data.

## Acknowledgements

We would like to thank Federico Faggin and Carver Mead for their good advice. Thanks to John Lazzaro who gave us a new version of Until, a graphics editor. We would also like to thank Steven Rosenberg and Bo Curry of Hewlett-Packard Laboratories for their suggestions and support.

## Footnotes

*Current address: Caltech Computer Graphics Group, Caltech 350-74, Pasadena, CA 92115

## References

Anderson, J., Mead, C., 1990, MOS Device for Long-Term Learning, U. S. Patent 4,935,702.

Anderson, J., Mead, C., Allen, T., Wall, M., 1992, Adaptable MOS Current Mirror, U. S. Patent 5,160,899.

Lee, Y., 1991, Handwritten Digit Recognition Using k Nearest-Neighbor, Radial Basis Function, and Backpropagation Neural Networks, *Neural Computation*, vol. 3, no. 3, 440–449.

Mead, C., Conway, L., 1980, Introduction to VLSI Systems, Addison-Wesley, Reading, MA.

Mead, C., 1989, Analog VLSI and Neural Systems, Addison-Wesley, Reading, MA.

Mead, C., Allen, T., Faggin, F., Anderson, J., 1992, Synaptic Element and Array, U. S. Patent 5,083,044.

Moody, J., Darken, C., 1989, Fast Learning in Networks of Locally-Tuned Processing Units, *Neural Computation*, vol. 1, no. 2, 281–294.

Platt, J., 1991, Learning by Combining Memorization and Gradient Descent, *In:* Advances in Neural Information Processing 3, Lippman, R., Moody, J.. Touretzky, D., eds., Morgan-Kaufmann, San Mateo, CA, 714–720.

Poggio, T., Girosi, F., 1990, Regularization Algorithms for Learning That Are Equivalent to Multilayer Networks, *Science*, vol. 247, 978–982.

Powell, M. J. D., 1987, Radial Basis Functions for Multivariable Interpolation: A Review, *In:* Algorithms for Approximation, J. C. Mason, M. G. Cox, eds., Clarendon Press, Oxford.